# Matrix Exponentiated Gradient Updates
# for On-line Learning and Bregman Projection

**Koji Tsuda**[*][†], **Gunnar Rätsch**[*][‡] **and Manfred K. Warmuth**[§]
[*]Max Planck Institute for Biological Cybernetics
Spemannstr. 38, 72076 Tübingen, Germany
[†]AIST CBRC, 2-43 Aomi, Koto-ku, Tokyo, 135-0064, Japan
[‡]Fraunhofer FIRST, Kekuléstr. 7, 12489 Berlin, Germany
[§]University of California at Santa Cruz
{koji.tsuda,gunnar.raetsch}@tuebingen.mpg.de,manfred@cse.ucsc.edu

## Abstract

We address the problem of learning a symmetric positive definite matrix. The central issue is to design parameter updates that preserve positive definiteness. Our updates are motivated with the *von Neumann* divergence. Rather than treating the most general case, we focus on two key applications that exemplify our methods: On-line learning with a simple square loss and finding a symmetric positive definite matrix subject to symmetric linear constraints. The updates generalize the Exponentiated Gradient (EG) update and AdaBoost, respectively: the parameter is now a symmetric positive definite matrix of trace one instead of a probability vector (which in this context is a diagonal positive definite matrix with trace one). The generalized updates use matrix logarithms and exponentials to preserve positive definiteness. Most importantly, we show how the analysis of each algorithm generalizes to the non-diagonal case. We apply both new algorithms, called the *Matrix Exponentiated Gradient* (MEG) update and *DefiniteBoost*, to learn a kernel matrix from distance measurements.

## 1 Introduction

Most learning algorithms have been developed to learn a *vector* of parameters from data. However, an increasing number of papers are now dealing with more structured parameters. More specifically, when learning a similarity or a distance function among objects, the parameters are defined as a *symmetric positive definite matrix* that serves as a kernel (e.g. [14, 11, 13]). Learning is typically formulated as a parameter updating procedure to optimize a *loss function*. The gradient descent update [6] is one of the most commonly used algorithms, but it is not appropriate when the parameters form a positive definite matrix, because the updated parameter is not necessarily positive definite. Xing et al. [14] solved this problem by always correcting the updated matrix to be positive. However no bound has been proven for this update-and-correction approach. In this paper, we introduce the *Matrix Exponentiated Gradient update* which works as follows: First, the matrix logarithm of the current parameter matrix is computed. Then a step is taken in the direction of the steepest descent. Finally, the parameter matrix is updated to the exponential of the modified log-matrix. Our update preserves symmetry and positive definiteness because the matrix exponential maps any symmetric matrix to a positive definite matrix.

Bregman divergences play a central role in the motivation and the analysis of *on-line learning algorithms* [5]. A learning problem is essentially defined by a loss function, and a divergence that measures the discrepancy between parameters. More precisely, the updates are motivated by minimizing the sum of the loss function and the Bregman divergence, where the loss function is multiplied by a positive learning rate. Different divergences lead to radically different updates [6]. For example, the gradient descent is derived from the squared Euclidean distance, and the exponentiated gradient from the Kullback-Leibler divergence. We use the *von Neumann* divergence (also called quantum relative entropy) for measuring the discrepancy between two positive definite matrices [8]. We derive a new *Matrix Exponentiated Gradient update* from this divergence (which is a Bregman divergence for positive definite matrices). Finally we prove *relative loss bounds* using the *von Neumann* divergence as a measure of progress.

Also the following related key problem has received a lot of attention recently [14, 11, 13]: Find a symmetric positive definite matrix that satisfies a number of symmetric linear inequality constraints. The new *DefiniteBoost* algorithm greedily chooses the most violated constraint and performs an approximated Bregman projection. In the diagonal case, we recover AdaBoost [9]. We also show how the convergence proof of AdaBoost generalizes to the non-diagonal case.

## 2   *von Neumann* Divergence or Quantum Relative Entropy

If F is a real convex differentiable function on the parameter domain (symmetric $d \times d$ positive definite matrices) and $\mathbf{f}(\mathbf{W}) := \nabla F(\mathbf{W})$, then the Bregman divergence between two parameters $\widetilde{\mathbf{W}}$ and $\mathbf{W}$ is defined as

$$\Delta_F(\widetilde{\mathbf{W}}, \mathbf{W}) = F(\widetilde{\mathbf{W}}) - F(\mathbf{W}) - \mathrm{tr}[(\widetilde{\mathbf{W}} - \mathbf{W})\mathbf{f}(\mathbf{W})].$$

When choosing $F(\mathbf{W}) = \mathrm{tr}(\mathbf{W} \log \mathbf{W} - \mathbf{W})$, then $\mathbf{f}(\mathbf{W}) = \log \mathbf{W}$ and the corresponding Bregman divergence becomes the *von Neumann* divergence [8]:

$$\Delta_F(\widetilde{\mathbf{W}}, \mathbf{W}) = \mathrm{tr}(\widetilde{\mathbf{W}} \log \widetilde{\mathbf{W}} - \widetilde{\mathbf{W}} \log \mathbf{W} - \widetilde{\mathbf{W}} + \mathbf{W}). \tag{1}$$

In this paper, we are primarily interested in the normalized case (when $\mathrm{tr}(\mathbf{W}) = 1$). In this case, the positive symmetric definite matrices are related to density matrices commonly used in Statistical Physics and the divergence simplifies to $\Delta_F(\widetilde{\mathbf{W}}, \mathbf{W}) = \mathrm{tr}(\widetilde{\mathbf{W}} \log \widetilde{\mathbf{W}} - \widetilde{\mathbf{W}} \log \mathbf{W})$.

If $\mathbf{W} = \sum_i \lambda_i \boldsymbol{v}_i \boldsymbol{v}_i^\top$ is our notation for the eigenvalue decomposition, then we can rewrite the normalized divergence as

$$\Delta_F(\widetilde{\mathbf{W}}, \mathbf{W}) = \sum_i \tilde{\lambda}_i \ln \tilde{\lambda}_i + \sum_{i,j} \tilde{\lambda}_i \ln \lambda_j (\tilde{\boldsymbol{v}}_i^\top \boldsymbol{v}_j)^2.$$

So this divergence quantifies the difference in the eigenvalues as well as the eigenvectors.

## 3   On-line Learning

In this section, we present a natural extension of the *Exponentiated Gradient* (EG) update [6] to an update for symmetric positive definite matrices.

At the $t$-th trial, the algorithm receives a symmetric instance matrix $\mathbf{X}_t \in \mathbb{R}^{d \times d}$. It then produces a prediction $\hat{y}_t = \mathrm{tr}(\mathbf{W}_t \mathbf{X}_t)$ based on the algorithm's current symmetric positive definite parameter matrix $\mathbf{W}_t$. Finally it incurs for instance[1] a quadratic loss $(\hat{y}_t - y_t)^2$,

and updates its parameter matrix $\mathbf{W}_t$. In the update we aim to solve the following problem:

$$\mathbf{W}_{t+1} = \operatorname{argmin}_{\mathbf{W}} \left( \Delta_F(\mathbf{W}, \mathbf{W}_t) + \eta(\operatorname{tr}(\mathbf{W}\mathbf{X}_t) - y_t)^2 \right), \qquad (2)$$

where the convex function F defines the Bregman divergence. Setting the derivative with respect to $\mathbf{W}$ to zero, we have

$$\mathbf{f}(\mathbf{W}_{t+1}) - \mathbf{f}(\mathbf{W}_t) + \eta\nabla[(\operatorname{tr}(\mathbf{W}_{t+1}\mathbf{X}_t) - y_t)^2] = 0. \qquad (3)$$

The update rule is derived by solving (3) with respect to $W_{t+1}$, but it is not solvable in closed form. A common way to avoid this problem is to approximate $\operatorname{tr}(\mathbf{W}_{t+1}\mathbf{X}_t)$ by $\operatorname{tr}(\mathbf{W}_t\mathbf{X}_t)$ [5]. Then, we have the following update:

$$\mathbf{W}_{t+1} = \mathbf{f}^{-1}(\mathbf{f}(\mathbf{W}_t) - 2\eta(\hat{y}_t - y_t)\mathbf{X}_t).$$

In our case, $F(\mathbf{W}) = \operatorname{tr}(\mathbf{W}\log\mathbf{W} - \mathbf{W})$ and thus $\mathbf{f}(\mathbf{W}) = \log\mathbf{W}$ and $\mathbf{f}^{-1}(\mathbf{W}) = \exp\mathbf{W}$. We also augment (2) with the constraint $\operatorname{tr}(\mathbf{W}) = 1$, leading to the following *Matrix Exponential Gradient (MEG) Update*:

$$\mathbf{W}_{t+1} = \frac{1}{Z_t} \exp(\log\mathbf{W}_t - 2\eta(\hat{y}_t - y_t)\mathbf{X}_t), \qquad (4)$$

where the normalization factor $Z_t$ is $\operatorname{tr}[\exp(\log\mathbf{W}_t - 2\eta(\hat{y}_t - y_t)\mathbf{X}_t)]$. Note that in the above update, the exponent $\log\mathbf{W}_t - 2\eta(\hat{y}_t - y_t)\mathbf{X}_t$ is an arbitrary symmetric matrix and the matrix exponential converts this matrix back into a symmetric positive definite matrix. A numerically stable version of the MEG update is given in Section 3.2.

## 3.1 Relative Loss Bounds

We now begin with the definitions needed for the relative loss bounds. Let $S = (\mathbf{X}_1, y_1), \ldots, (\mathbf{X}_T, y_T)$ denote a sequence of examples, where the instance matrices $\mathbf{X}_t \in \mathbb{R}^{d \times d}$ are symmetric and the labels $y_t \in \mathbb{R}$. For any symmetric positive semi-definite matrix $\mathbf{U}$ with $\operatorname{tr}(\mathbf{U}) = 1$, define its total loss as $L_{\mathbf{U}}(S) = \sum_{t=1}^{T}(\operatorname{tr}(\mathbf{U}\mathbf{X}_t) - y_t)^2$. The total loss of the on-line algorithm is $L_{MEG}(S) = \sum_{t=1}^{T}(\operatorname{tr}(\mathbf{W}_t\mathbf{X}_t) - y_t)^2$. We prove a bound on the *relative loss* $L_{MEG}(S) - L_{\mathbf{U}}(S)$ that holds for any $\mathbf{U}$. The proof generalizes a similar bound for the Exponentiated Gradient update (Lemmas 5.8 and 5.9 of [6]). The relative loss bound is derived in two steps: Lemma 3.1 bounds the relative loss for an individual trial and Lemma 3.2 for a whole sequence (Proofs are given in the full paper).

**Lemma 3.1** *Let $\mathbf{W}_t$ be any symmetric positive definite matrix. Let $\mathbf{X}_t$ be any symmetric matrix whose smallest and largest eigenvalues satisfy $\lambda^{\max} - \lambda^{\min} \leq r$. Assume $\mathbf{W}_{t+1}$ is produced from $\mathbf{W}_t$ by the MEG update and let $\mathbf{U}$ be any symmetric positive semi-definite matrix. Then for any constants $a$ and $b$ such that $0 < a \leq 2b/(2 + r^2 b)$ and any learning rate $\eta = 2b/(2 + r^2 b)$, we have*

$$a(y_t - \operatorname{tr}(\mathbf{W}_t\mathbf{X}_t))^2 - b(y_t - \operatorname{tr}(\mathbf{U}\mathbf{X}_t))^2 \leq \Delta(\mathbf{U}, \mathbf{W}_t) - \Delta(\mathbf{U}, \mathbf{W}_{t+1}) \qquad (5)$$

In the proof, we use the Golden-Thompson inequality [3], i.e., $\operatorname{tr}[\exp(\mathbf{A} + \mathbf{B})] \geq \operatorname{tr}[\exp(\mathbf{A})\exp(\mathbf{B})]$ for symmetric matrices $\mathbf{A}$ and $\mathbf{B}$. We also needed to prove the following generalization of Jensen's inequality to matrices: $\exp(\rho_1\mathbf{A} + \rho_2(\mathbf{I} - \mathbf{A})) \leq \exp(\rho_1)\mathbf{A} + \exp(\rho_2)(\mathbf{I} - \mathbf{A})$ for finite $\rho_1, \rho_2 \in \mathbb{R}$ and any symmetric matrix $\mathbf{A}$ with $0 < \mathbf{A} \leq \mathbf{I}$. These two key inequalities will also be essential for the analysis of *Definite-Boost* in the next section.

**Lemma 3.2** *Let $\mathbf{W}_1$ and $\mathbf{U}$ be arbitrary symmetric positive definite initial and comparison matrices, respectively. Then for any $c$ such that $\eta = 2c/(r^2(2 + c))$,*

$$L_{MEG}(S) \leq \left(1 + \frac{c}{2}\right)L_{\mathbf{U}}(S) + \left(\frac{1}{2} + \frac{1}{c}\right)r^2\Delta(\mathbf{U}, \mathbf{W}_1). \qquad (6)$$

**Proof** For the maximum tightness of (5), $a$ should be chosen as $a = \eta = 2b/(2 + r^2 b)$. Let $b = c/r^2$, and thus $a = 2c/(r^2(2 + c))$. Then (5) is rewritten as

$$\frac{2c}{2 + c}(y_t - \text{tr}(\mathbf{W}_t \mathbf{X}_t))^2 - c(y_t - \text{tr}(\mathbf{U}\mathbf{X}_t))^2 \leq r^2(\Delta(\mathbf{U}, \mathbf{W}_t) - \Delta(\mathbf{U}, \mathbf{W}_{t+1}))$$

Adding the bounds for $t = 1, \cdots, T$, we get

$$\frac{2c}{2 + c}L_{MEG}(S) - cL_{\mathbf{U}}(S) \leq r^2(\Delta(\mathbf{U}, \mathbf{W}_1) - \Delta(\mathbf{U}, \mathbf{W}_{t+1})) \leq r^2 \Delta(\mathbf{U}, \mathbf{W}_1),$$

which is equivalent to (6). ∎

Assuming $L_{\mathbf{U}}(S) \leq \ell_{\max}$ and $\Delta(\mathbf{U}, \mathbf{W}_1) \leq d_{\max}$, the bound (6) is tightest when $c = r\sqrt{2d_{\max}/\ell_{\max}}$. Then we have $L_{MEG}(S) - L_{\mathbf{U}}(S) \leq r\sqrt{2\ell_{\max}d_{\max}} + \frac{r^2}{2}\Delta(\mathbf{U}, \mathbf{W}_1)$.

### 3.2 Numerically stable MEG update

The MEG update is numerically unstable when the eigenvalues of $\mathbf{W}_t$ are around zero. However we can "unwrap" $\mathbf{W}_{t+1}$ as follows:

$$\mathbf{W}_{t+1} = \frac{1}{\tilde{Z}_t}\exp(c_t\mathbf{I} + \log\mathbf{W}_1 - 2\eta\sum_{s=1}^{t}(\hat{y}_s - y_s)\mathbf{X}_s), \tag{7}$$

where the constant $\tilde{Z}_t$ normalizes the trace of $\mathbf{W}_{t+1}$ to one. As long as the eigen values of $\mathbf{W}_1$ are not too small then the computation of $\log\mathbf{W}_t$ is stable. Note that the update is independent of the choice of $c_t \in \mathbb{R}$. We incrementally maintain an eigenvalue decomposition of the matrix in the exponent ($O(n^3)$ per iteration):

$$\mathbf{V}_t\mathbf{\Lambda}_t\mathbf{V}_t^T = c_t\mathbf{I} + \log\mathbf{W}_1 - 2\eta\sum_{s=1}^{t}(\hat{y}_s - y_s)\mathbf{X}_s),$$

where the constant $c_t$ is chosen so that the maximum eigenvalue of the above is zero. Now $\mathbf{W}_{t+1} = \mathbf{V}_t\exp(\mathbf{\Lambda}_t)\mathbf{V}_t^T/\text{tr}(\exp(\mathbf{\Lambda}_t))$.

## 4 Bregman Projection and *DefiniteBoost*

In this section, we address the following Bregman projection problem[2]

$$\mathbf{W}^* = \text{argmin}_{\mathbf{W}}\,\Delta_{\mathcal{F}}(\mathbf{W}, \mathbf{W}_1),\ \text{tr}(\mathbf{W}) = 1, \text{tr}(\mathbf{W}\mathbf{C}_j) \leq 0,\ \text{for } j = 1, \dots, n, \tag{8}$$

where the symmetric positive definite matrix $\mathbf{W}_1$ of trace one is the initial parameter matrix, and $\mathbf{C}_1, \dots, \mathbf{C}_n$ are arbitrary symmetric matrices. Prior knowledge about $\mathbf{W}$ is encoded in the constraints, and the matrix closest to $\mathbf{W}_1$ is chosen among the matrices satisfying all constraints. Tsuda and Noble [13] employed this approach for learning a kernel matrix among graph nodes, and this method can be potentially applied to learn a kernel matrix in other settings (e.g. [14, 11]).

The problem (8) is a projection of $\mathbf{W}_1$ to the intersection of convex regions defined by the constraints. It is well known that the Bregman projection into the intersection of convex regions can be solved by sequential projections to each region [1]. In the original papers only asymptotic convergence was shown. More recently a connection [4, 7] was made to the AdaBoost algorithm which has an improved convergence analysis [2, 9]. We generalize the latter algorithm and its analysis to symmetric positive definite matrices and call the new algorithm *DefiniteBoost*. As in the original setting, only *approximate* projections (Figure 1) are required to show fast convergence.

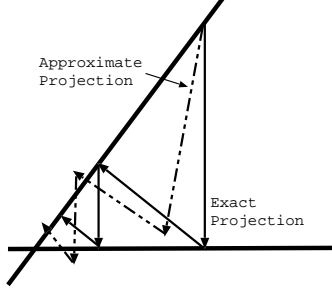

Figure 1: In (exact) Bregman projections, the intersection of convex sets (i.e., two lines here) is found by iterating projections to each set. We project only approximately, so the projected point does not satisfy the current constraint. Nevertheless, global convergence to the optimal solution is guaranteed via our proofs.

Before presenting the algorithm, let us derive the dual problem of (8) by means of Lagrange multipliers $\boldsymbol{\gamma}$,

$$\boldsymbol{\gamma}^* = \mathrm{argmin}_{\boldsymbol{\gamma}} \log\left(\mathrm{tr}\left[\exp(\log \mathbf{W}_1 - \sum_{j=1}^{n} \gamma_j \mathbf{C}_j)\right]\right), \quad \gamma_j \geq 0. \tag{9}$$

See [13] for a detailed derivation of the dual problem. When (8) is feasible, the optimal solution is described as $\mathbf{W}^* = \frac{1}{Z(\boldsymbol{\gamma}^*)} \exp(\log \mathbf{W}_1 - \sum_{j=1}^{n} \gamma_j^* \mathbf{C}_j)$, where $Z(\boldsymbol{\gamma}^*) = \mathrm{tr}[\exp(\log \mathbf{W}_1 - \sum_{j=1}^{n} \gamma_j^* \mathbf{C}_j)]$.

### 4.1 Exact Bregman Projections

First, let us present the exact Bregman projection algorithm to solve (8). We start from the initial parameter $\mathbf{W}_1$. At the $t$-th step, the most unsatisfied constraint is chosen, $j_t = \mathrm{argmax}_{j=1,\cdots,n} \mathrm{tr}(\mathbf{W}_t \mathbf{C}_j)$. Let us use $\mathbf{C}_t$ as the short notation for $\mathbf{C}_{j_t}$. Then, the following Bregman projection with respect to the chosen constraint is solved.

$$\mathbf{W}_{t+1} = \mathrm{argmin}_{\mathbf{W}} \Delta(\mathbf{W}, \mathbf{W}_t), \quad \mathrm{tr}(\mathbf{W}) = 1, \mathrm{tr}(\mathbf{W}\mathbf{C}_t) \leq 0. \tag{10}$$

By means of a Lagrange multiplier $\alpha$, the dual problem is described as

$$\alpha_t = \mathrm{argmin}_{\alpha} \mathrm{tr}[\exp(\log \mathbf{W}_t - \alpha \mathbf{C}_t)], \quad \alpha \geq 0. \tag{11}$$

Using the solution of the dual problem, $W_t$ is updated as

$$\mathbf{W}_{t+1} = \frac{1}{Z_t(\alpha_t)} \exp(\log \mathbf{W}_t - \alpha_t \mathbf{C}_t) \tag{12}$$

where the normalization factor is $Z_t(\alpha_t) = \mathrm{tr}[\exp(\log \mathbf{W}_t - \alpha_t \mathbf{C}_t)]$. Note that we can use the same numerically stable update as in the previous section.

### 4.2 Approximate Bregman Projections

The solution of (11) cannot be obtained in closed form. However, one can use the following approximate solution:

$$\alpha_t = \frac{1}{\lambda_t^{\max} - \lambda_t^{\min}} \log\left(\frac{1 + r_t/\lambda_t^{\max}}{1 + r_t/\lambda_t^{\min}}\right), \tag{13}$$

when the eigenvalues of $\mathbf{C}_t$ lie in the interval $[\lambda_t^{\min}, \lambda_t^{\max}]$ and $r_t = \mathrm{tr}(\mathbf{W}_t \mathbf{C}_t)$. Since the most unsatisfied constraint is chosen, $r_t \geq 0$ and thus $\alpha_t \geq 0$. Although the projection is done only approximately,[3] the convergence of the dual objective (9) can be shown using the following upper bound.

**Theorem 4.1** *The dual objective* (9) *is bounded as*

$$\mathrm{tr}\left[\exp\left(\log\mathbf{W}_1 - \sum_{j=1}^{n}\gamma_j\mathbf{C}_j\right)\right] \le \prod_{t=1}^{T}\rho(r_t) \tag{14}$$

$$\textit{where } \rho(r_t) = \left(1 - \frac{r_t}{\lambda_t^{\max}}\right)^{\frac{\lambda_t^{\max}}{\lambda_t^{\max}-\lambda_t^{\min}}} \left(1 - \frac{r_t}{\lambda_t^{\min}}\right)^{\frac{-\lambda_t^{\min}}{\lambda_t^{\max}-\lambda_t^{\min}}}.$$

The dual objective is monotonically decreasing, because $\rho(r_t) \le 1$. Also, since $r_t$ corresponds to the maximum value among all constraint violations $\{r_j\}_{j=1}^{n}$, we have $\rho(r_t) = 1$ only if $r_t = 0$. Thus the dual objective continues to decrease until all constraints are satisfied.

### 4.3 Relation to Boosting

When all matrices are diagonal, the DefiniteBoost degenerates to AdaBoost [9]: Let $\{\boldsymbol{x}_i, y_i\}_{i=1}^{d}$ be the training samples, where $\boldsymbol{x}_i \in \mathbb{R}^m$ and $y_i \in \{-1, 1\}$. Let $h_1(x), \ldots, h_n(x) \in [-1, 1]$ be the weak hypotheses. For the $j$-th hypothesis $h_j(x)$, let us define $\mathbf{C}_j = \mathrm{diag}(y_1 h_j(x_1), \ldots, y_d h_j(x_d))$. Since $|yh_j(x)| \le 1$, $\lambda_t^{\max / \min} = \pm 1$ for any $t$. Setting $\mathbf{W}_1 = \mathbf{I}/d$, the dual objective (14) is rewritten as

$$\frac{1}{d}\sum_{i=1}^{d}\exp\left(-y_i\sum_{j=1}^{n}\gamma_j h_j(x_i)\right),$$

which is equivalent to the exponential loss function used in AdaBoost. Since $\mathbf{C}_j$ and $\mathbf{W}_1$ are diagonal, the matrix $W_t$ stays diagonal after the update. If $w_{ti} = [\mathbf{W}_t]_{ii}$, the updating formula (12) becomes the AdaBoost update: $w_{t+1,i} = w_{ti}\exp(-\alpha_t y_i h_t(x_i))/\mathrm{Z}_t(\alpha_t)$. The approximate solution of $\alpha_t$ (13) is described as $\alpha_t = \frac{1}{2}\log\frac{1+r_t}{1-r_t}$, where $r_t$ is the weighted training error of the $t$-th hypothesis, i.e. $r_t = \sum_{i=1}^{d}w_{ti}y_i h_t(x_i)$.

## 5 Experiments on Learning Kernels

In this section, our technique is applied to learning a kernel matrix from a set of distance measurements. This application is not on-line per se, but it shows nevertheless that the theoretical bounds can be reasonably tight on natural data.

When $\mathbf{K}$ is a $d \times d$ kernel matrix among $d$ objects, then the $K_{ij}$ characterizes the similarity between objects $i$ and $j$. In the feature space, $K_{ij}$ corresponds to the inner product between object $i$ and $j$, and thus the Euclidean distance can be computed from the entries of the kernel matrix [10]. In some cases, the kernel matrix is not given explicitly, but only a set of distance measurements is available. The data are represented either as (i) quantitative distance values (e.g., the distance between $i$ and $j$ is 0.75), or (ii) qualitative evaluations (e.g., the distance between $i$ and $j$ is small) [14, 13]. Our task is to obtain a positive definite kernel matrix which fits well to the given distance data.

**On-line kernel learning**   In the first experiment, we consider the on-line learning scenario in which only one distance example is shown to the learner at each time step. The distance example at time $t$ is described as $\{a_t, b_t, y_t\}$, which indicates that the squared Euclidean distance between objects $a_t$ and $b_t$ is $y_t$. Let us define a time-developing sequence of kernel matrices as $\{\mathbf{W}_t\}_{t=1}^{T}$, and the corresponding points in the feature space as $\{\boldsymbol{x}_{ti}\}_{i=1}^{d}$ (i.e. $[\mathbf{W}_t]_{ab} = \boldsymbol{x}_{ta}^{\top}\boldsymbol{x}_{tb}$). Then, the total loss incurred by this sequence is

$$\sum_{t=1}^{T}\left(\|\boldsymbol{x}_{ta_t} - \boldsymbol{x}_{tb_t}\|^2 - y_t\right)^2 = \sum_{t=1}^{T}(\mathrm{tr}(\mathbf{W}_t\mathbf{X}_t) - y_t)^2,$$

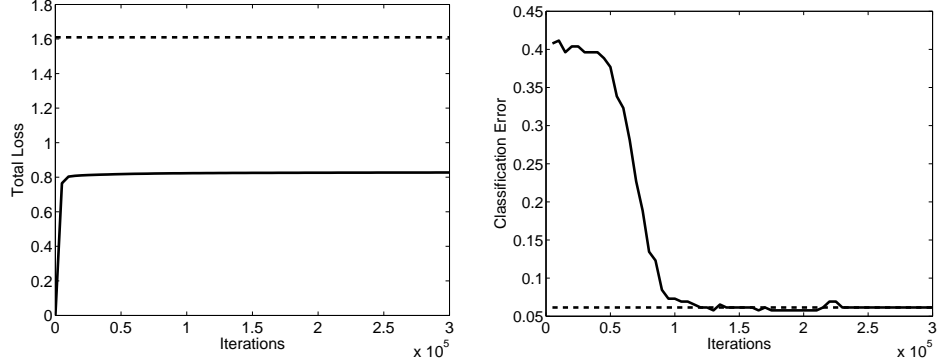

Figure 2: Numerical results of on-line learning. (Left) total loss against the number of iterations. The dashed line shows the loss bound. (Right) classification error of the nearest neighbor classifier using the learned kernel. The dashed line shows the error by the target kernel.

where $\mathbf{X}_t$ is a symmetric matrix whose $(a_t, a_t)$ and $(b_t, b_t)$ elements are 0.5, $(a_t, b_t)$ and $(b_t, a_t)$ elements are -0.5, and all the other elements are zero. We consider a controlled experiment in which the distance examples are created from a known *target kernel matrix*. We used a $52 \times 52$ kernel matrix among gyrB proteins of bacteria $(d = 52)$. This data contains three bacteria species (see [12] for details). Each distance example is created by randomly choosing one element of the target kernel. The initial parameter was set as $\mathbf{W}_1 = \mathbf{I}/d$. When the comparison matrix $U$ is set to the target matrix, $L_U(S) = 0$ and $\ell_{\max} = 0$, because all the distance examples are derived from the target matrix. Therefore we choose learning rate $\eta = 2$, which minimizes the relative loss bound of Lemma 3.2. The total loss of the kernel matrix sequence obtained by the matrix exponential update is shown in Figure 2 (left). In the plot, we have also shown the relative loss bound. The bound seems to give a reasonably tight performance guarantee—it is about twice the actual total loss. To evaluate the learned kernel matrix, the prediction accuracy of bacteria species by the nearest neighbor classifier is calculated (Figure 2, right), where the 52 proteins are randomly divided into 50% training and 50% testing data. The value shown in the plot is the test error averaged over 10 different divisions. It took a large number of iterations $(\sim 2 \times 10^5)$ for the error rate to converge to the level of the target kernel. In practice one can often increase the learning rate for faster convergence, but here we chose the small rate suggested by our analysis to check the tightness of the bound.

**Kernel learning by Bregman projection**   Next, let us consider a batch learning scenario where we have a set of qualitative distance evaluations (i.e. inequality constraints). Given $n$ pairs of similar objects $\{a_j, b_j\}_{j=1}^n$, the inequality constraints are constructed as $\|\boldsymbol{x}_{a_j} - \boldsymbol{x}_{b_j}\| \leq \gamma, j = 1, \ldots, n$, where $\gamma$ is a predetermined constant. If $\mathbf{X}_j$ is defined as in the previous section and $\mathbf{C}_j = \mathbf{X}_j - \gamma\mathbf{I}$, the inequalities are then rewritten as $\mathrm{tr}(\mathbf{W}\mathbf{C}_j) \leq 0, j = 1, \ldots, n$. The largest and smallest eigenvalues of any $\mathbf{C}_j$ are $1 - \gamma$ and $-\gamma$, respectively. As in the previous section, distance examples are generated from the target kernel matrix between gyrB proteins. Setting $\gamma = 0.2/d$, we collected all object pairs whose distance in the feature space is less than $\gamma$ to yield 980 inequalities $(n = 980)$. Figure 3 (left) shows the convergence of the dual objective function as proven in Theorem 4.1. The convergence was much faster than the previous experiment, because, in the batch setting, one can choose the most unsatisfied constraint, and optimize the step size as well. Figure 3 (right) shows the classification error of the nearest neighbor classifier. As opposed to the previous experiment, the error rate is higher than that of the target kernel matrix, because substantial amount of information is lost by the conversion to inequality constraints.

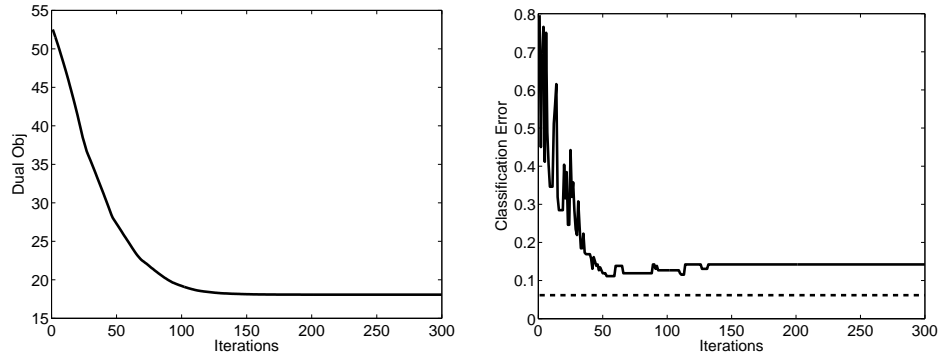

Figure 3: Numerical results of Bregman projection. (Left) convergence of the dual objective function. (Right) classification error of the nearest neighbor classifier using the learned kernel.

## 6 Conclusion

We motivated and analyzed a new update for symmetric positive matrices using the *von Neumann* divergence. We showed that the standard bounds for on-line learning and Boosting generalize to the case when the parameters are a symmetric positive definite matrix (of trace one) instead of a probability vector. As in quantum physics, the eigenvalues act as probabilities.

**Acknowledgment** We would like to thank B. Schölkopf, M. Kawanabe, J. Liao and W.S. Noble for fruitful discussions. M.W. was supported by NSF grant CCR 9821087 and UC Discovery grant LSIT02-10110. K.T. and G.R. gratefully acknowledge partial support from the PASCAL Network of Excellence (EU #506778). Part of this work was done while all three authors were visiting the National ICT Australia in Canberra.

## Footnotes

[1]For the sake of simplicity, we use the simple quadratic loss: $L_t(\mathbf{W}) = (\mathrm{tr}(\mathbf{X}_t \mathbf{W}) - y_t)^2$. For the general update, the gradient $\nabla L_t(\mathbf{W}_t)$ is exponentiated in the update (4) and this gradient must be symmetric. Following [5], more general loss functions (based on Bregman divergences) are amenable to our techniques.

[2]Note that if $\eta$ is large then the on-line update (2) becomes a Bregman projection subject to a single equality constraint $\text{tr}(\mathbf{W}\mathbf{X}_t) = y_t$.

[3]The approximate Bregman projection (with $\alpha_t$ as in (13) can also be motivated as an online algorithm based on an entropic loss and learning rate one (following Section 3 and [4]).

## References

[1] L.M. Bregman. Finding the common point of convex sets by the method of successive projections. *Dokl. Akad. Nauk SSSR*, 165:487–490, 1965.

[2] Y. Freund and R.E. Schapire. A decision-theoretic generalization of on-line learning and an application to boosting. *Journal of Computer and System Sciences*, 55(1):119–139, 1997.

[3] S. Golden. Lower bounds for the Helmholtz function. *Phys. Rev.*, 137:B1127–B1128, 1965.

[4] J. Kivinen and M. K. Warmuth. Boosting as entropy projection. In *Proc. 12th Annu. Conference on Comput. Learning Theory*, pages 134–144. ACM Press, New York, NY, 1999.

[5] J. Kivinen and M. K. Warmuth. Relative loss bounds for multidimensional regression problems. *Machine Learning*, 45(3):301–329, 2001.

[6] J. Kivinen and M.K. Warmuth. Exponentiated gradient versus gradient descent for linear predictors. *Information and Computation*, 132(1):1–63, 1997.

[7] J. Lafferty. Additive models, boosting, and inference for generalized divergences. In *Proc. 12th Annu. Conf. on Comput. Learning Theory*, pages 125–133, New York, NY, 1999. ACM Press.

[8] M.A. Nielsen and I.L. Chuang. *Quantum Computation and Quantum Information*. Cambridge University Press, 2000.

[9] R.E. Schapire and Y. Singer. Improved boosting algorithms using confidence-rated predictions. *Machine Learning*, 37:297–336, 1999.

[10] B. Schölkopf and A. J. Smola. *Learning with Kernels*. MIT Press, Cambridge, MA, 2002.

[11] I.W. Tsang and J.T. Kwok. Distance metric learning with kernels. In *Proceedings of the International Conference on Artificial Neural Networks (ICANN'03)*, pages 126–129, 2003.

[12] K. Tsuda, S. Akaho, and K. Asai. The em algorithm for kernel matrix completion with auxiliary data. *Journal of Machine Learning Research*, 4:67–81, May 2003.

[13] K. Tsuda and W.S. Noble. Learning kernels from biological networks by maximizing entropy. *Bioinformatics*, 2004. to appear.

[14] E.P. Xing, A.Y. Ng, M.I. Jordan, and S. Russell. Distance metric learning with application to clustering with side-information. In S. Thrun S. Becker and K. Obermayer, editors, *Advances in Neural Information Processing Systems 15*, pages 505–512. MIT Press, Cambridge, MA, 2003.